# Transforming Neural-Net Output Levels to Probability Distributions

**John S. Denker and Yann leCun**
AT&T Bell Laboratories
Holmdel, NJ 07733

## Abstract

(1) The outputs of a typical multi-output classification network do not satisfy the axioms of probability; probabilities should be positive and sum to one. This problem can be solved by treating the trained network as a *preprocessor* that produces a *feature vector* that can be further processed, for instance by classical statistical estimation techniques. (2) We present a method for computing the first two moments of the probability distribution indicating the range of outputs that are consistent with the input and the training data. It is particularly useful to combine these two ideas: we implement the ideas of section 1 using Parzen windows, where the shape and relative size of each window is computed using the ideas of section 2. This allows us to make contact between important theoretical ideas (e.g. the ensemble formalism) and practical techniques (e.g. back-prop). Our results also shed new light on and generalize the well-known "softmax" scheme.

## 1  Distribution of Categories in Output Space

In many neural-net applications, it is crucial to produce a set of $C$ numbers that serve as estimates of the probability of $C$ mutually exclusive outcomes. For example, in speech recognition, these numbers represent the probability of $C$ different phonemes; the probabilities of successive segments can be combined using a Hidden Markov Model. Similarly, in an Optical Character Recognition ("OCR") application, the numbers represent $C$ possible characters. Probability information for the "best guess" category (and probable runner-up categories) is combined with context, cost information, etcetera, to produce recognition of multi-character strings.

According to the axioms of probability, these $C$ numbers should be constrained to be positive and sum to one. We find that rather than modifying the network architecture and/or training algorithm to satisfy this constraint directly, it is advantageous to use a network without the probabilistic constraint, followed by a statistical post-processor. Similar strategies have been discussed before, e.g. (Fogelman, 1990).

The obvious starting point is a network with $C$ output units. We can train the network with targets that obey the probabilistic constraint, e.g. the target for category "0" is $[1, 0, 0, \cdots]$, the target for category "1" is $[0, 1, 0, \cdots]$, etcetera. This would not, alas, guarantee that the *actual* outputs would obey the constraint. Of course, the actual outputs can always be shifted and normalized to meet the requirement; one of the goals of this paper is to understand the best way to perform such a transformation. A more sophisticated idea would be to construct a network that had such a transformation (e.g. softmax (Bridle, 1990; Rumelhart, 1989)) "built in" even during training. We tried this idea and discovered numerous difficulties, as discussed in (Denker and leCun, 1990).

The most principled solution is simply to collect statistics on the trained network. Figures 1 and 2 are scatter plots of output from our OCR network (Le Cun et al., 1990) that was trained to recognize the digits "0" through "9". In the first figure, the outputs tend to cluster around the target vectors [the points $(T^-, T^+)$ and $(T^+, T^-)$], and even though there are a few stragglers, decision regions can be found that divide the space into a high-confidence "0" region, a high-confidence "1" region, and a quite small "rejection" region. In the other figure, it can be seen that the "3 versus 5" separation is very challenging.

In all cases, the plotted points indicate the output of the network when the input image is taken from a special "calibration" dataset $\mathcal{L}$ that is distinct both from the training set $\mathcal{M}$ (used to train the network) and from the testing set $\mathcal{G}$ (used to evaluate the generalization performance of the final, overall system).

This sort of analysis is applicable to a wide range of problems. The architecture of the neural network (or other adaptive system) should be chosen to suit the problem in each case. The network should then be trained using standard techniques. The hope is that the output will constitute a *sufficent statistic*.

Given enough training data, we could use a standard statistical technique such as Parzen windows (Duda and Hart, 1973) to estimate the probability density in output space. It is then straightforward to take an unknown input, calculate the corresponding output vector $O$, and then estimate the probability that it belongs to each class, according to the density of points of category $c$ "at" location $O$ in the scatter plot.

We note that methods such as Parzen windows tend to fail when the number of dimensions becomes too large, because it is exponentially harder to estimate probability densities in high-dimensional spaces; this is often referred to as "the curse of dimensionality" (Duda and Hart, 1973). Since the number of output units (typically 10 in our OCR network) is much smaller than the number of input units (typically 400) the method proposed here has a tremendous advantage compared to classical statistical methods applied directly to the input vectors. This advantage is increased by the fact that the distribution of points in network-output space is much more regular than the distribution in the original space.

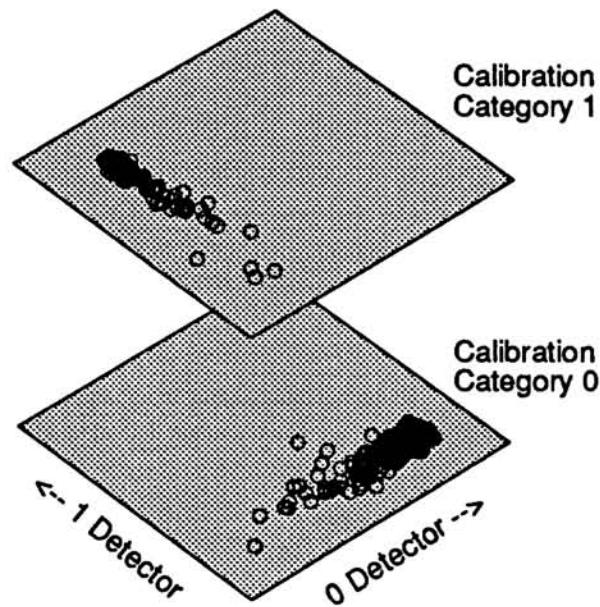

Figure 1: Scatter Plot: Category 1 versus 0

One axis in each plane represents the activation level of output unit $j=0$, while the other axis represents activation level of output unit $j=1$; the other 8 dimensions of output space are suppressed in this projection. Points in the upper and lower plane are, respectively, assigned category "1" and "0" by the calibration set. The clusters appear elongated because there are so many ways that an item can be neither a "1" nor a "0." This figure contains over 500 points; the cluster centers are heavily overexposed.

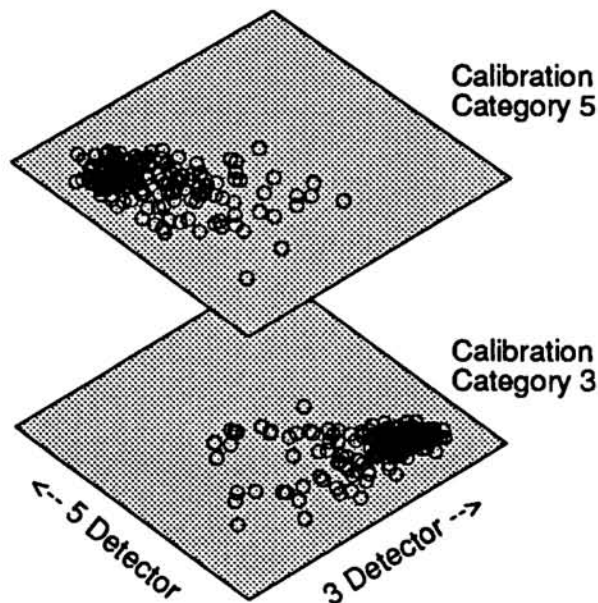

Figure 2: Scatter Plot: Category 5 versus 3

This is the same as the previous figure except for the choice of data points and projection axes.

## 2    Output Distribution for a Particular Input

The purpose of this section is to discuss the effect that limitations in the quantity and/or quality of training data have on the reliability of neural-net outputs. Only an outline of the argument can be presented here; details of the calculation can be found in (Denker and leCun, 1990). This section does not use the ideas developed in the previous section; the two lines of thought will converge in section 3. The calculation proceeds in two steps: (1) to calculate the range of weight values consistent with the training data, and then (2) to calculate the sensitivity of the output to uncertainty in weight space. The result is a network that not only produces a "best guess" output, but also an "error bar" indicating the confidence interval around that output.

The best formulation of the problem is to imagine that the input-output relation of the network is given by a probability distribution $P(O, I)$ [rather than the usual function $O = f(I)$] where $I$ and $O$ represent the input vector and output vector respectively. For any specific input pattern, we get a probability distribution $P_{OI}(O|I)$, which can be thought of as a histogram describing the probability of various output values.

Even for a definite input $I$, the output will be probabilistic, because there is never enough information in the training set to determine the precise value of the weight vector $W$. Typically there are non-trivial error bars on the training data. Even when the training data is absolutely noise-free (e.g. when it is generated by a mathematical function on a discrete input space (Denker et al., 1987)) the output can still be uncertain if the network is underdetermined; the uncertainty arises from lack of data quantity, not quality. In the real world one is faced with *both* problems: less than enough data to (over)determine the network, and less than complete confidence in the data that does exist.

We assume we have a handy method (e.g. back-prop) for finding a (local) minimum $\bar{W}$ of the loss function $E(W)$. A second-order Taylor expansion should be valid in the vicinity of $\bar{W}$. Since the loss function $E$ is an additive function of training data, and since probabilities are multiplicative, it is not surprising that the likelihood of a weight configuration is an exponential function of the loss (Tishby, Levin and Solla, 1989). Therefore the probability can be modelled locally as a multidimensional gaussian centered at $\bar{W}$; to a reasonable (Denker and leCun, 1990) approximation the probability is proportional to:

$$\rho_m(W) = \rho_0(W) \exp[-\beta \sum_i h_{ii}(W_i - \bar{W}_i)^2/2] \tag{1}$$

where $h$ is the second derivative of the loss (the Hessian), $\beta$ is a scale factor that determines our overall confidence in the training data, and $\rho_0$ expresses any information we have about prior probabilities. The sums run over the dimensions of parameter space. The width of this gaussian describes the range of networks in the ensemble that are reasonably consistent with the training data.

Because we have a probability distribution on $W$, the expression $O = f_W(I)$ gives a probability distribution on outputs $O$, even for fixed inputs $I$. We find that the most probable output $\bar{O}$ corresponds to the most probable parameters $\bar{W}$. This unsurprising result indicates that we are on the right track.

We next would like to know what *range* of output values correspond to the allowed range of parameter values. We start by calculating the sensitivity of the output $O = f_W(I)$ to changes in $W$ (holding the input $I$ fixed). For each output unit $j$, the derivative of $O_j$ with respect to $W$ can be evaluated by a straightforward modification of the usual back-prop algorithm.

Our distribution of output values also has a second moment, which is given by a surprisingly simple expression:

$$\sigma_j^2 = \langle (O_j - \bar{O}_j)^2 \rangle_{\rho_m} = \sum_i \frac{\gamma_{j,i}^2}{\beta h_{ii}} \tag{2}$$

where $\gamma_{j,i}$ denotes the gradient of $O_j$ with respect to $W_i$. We now have the first two moments of the output probability distribution ($\bar{O}$ and $\sigma$); we could calculate more if we wished.

It is reasonable to expect that the weighted sums (*before* the squashing function) at the last layer of our network are approximately normally distributed, since they are sums of random variables. If the output units are arranged to be reasonably linear, the output distribution is then given by

$$P_{jI}(O_j|I) = \mathcal{N}[\bar{O}_j, \sigma_j](O_j) \tag{3}$$

where $\mathcal{N}$ is the conventional Normal (Gaussian) distribution with given mean and variance, and where $\bar{O}$ and $\sigma$ depend on $I$. For multiple output units, we must consider the joint probability distribution $P_{OI}(O|I)$. If the different output units' distributions are independent, $P_{OI}$ can be factored:

$$P_{OI}(O|I) = \prod_j P_{jI}(O_j|I) \tag{4}$$

We have achieved the goal of this section: a formula describing a distribution of outputs consistent with a given input. This is a much fancier statement than the vanilla network's statement that $\bar{O}$ is "the" output. For a network that is not underdetermined, in the limit $\beta \to \infty$, $P_{OI}$ becomes a $\delta$ function located at $\bar{O}$, so our formalism contains the vanilla network as a special case. For general $\beta$, the region where $P_{OI}$ is large constitutes a "confidence region" of size proportional to the fuzziness $1/\beta$ of the data and to the degree to which the network is underdetermined.

Note that algorithms exist (Becker and Le Cun, 1989), (Le Cun, Denker and Solla, 1990) for calculating $\gamma$ and $h$ very efficiently — the time scales linearly with the time of calculation of $\bar{O}$. Equation 4 is remarkable in that it makes contact between important theoretical ideas (e.g. the ensemble formalism) and practical techniques (e.g. back-prop).

## 3    Combining the Distributions

Our main objective is an expression for $P(c|I)$, the probability that input $I$ should be assigned category $c$. We get it by combining the idea that elements of the calibration set $\mathcal{L}$ are scattered in output space (section 1) with the idea that the network output for each such element is uncertain because the network is under-determined (section 2). We can then draw a scatter plot in which the calibration

data is represented not by zero-size *points* but by *distributions* in output space. One can imagine each element of $\mathcal{L}$ as covering the area spanned by its "error bars" of size $\sigma$ as given by equation 2. We can then calculate $P(c|I)$ using ideas analogous to Parzen windows, with the advantage that the shape and relative size of each window is calculated, not assumed. The answer comes out to be:

$$P(c|I) = \int \frac{\sum_{l\in\mathcal{L}^c} P_{OI}(O|I^l)}{\sum_{l\in\mathcal{L}} P_{OI}(O|I^l)} P_{OI}(O|I)\, dO \tag{5}$$

where we have introduced $\mathcal{L}^c$ to denote the subset of $\mathcal{L}$ for which the assigned category is $c$. Note that $P_{OI}$ (given by equation 4) is being used in two ways in this formula: to calibrate the statistical postprocessor by summing over the elements of $\mathcal{L}$, and also to calculate the fate of the input $I$ (an element of the testing set).

Our result can be understood by analogy to Parzen windows, although it differs from the standard Parzen windows scheme in two ways. First, it is pleasing that we have a way of calculating the shape and relative size of the windows, namely $P_{OI}$. Secondly, after we have summed the windows over the calibration set $\mathcal{L}$, the standard scheme would probe each window at the single point $\bar{O}$; our expression (equation 5) accounts for the fact that the network's response to the testing input $I$ is blurred over a region given by $P_{OI}(O|I)$ and calls for a convolution.

## Correspondence with Softmax

We were not surprised that, in suitable limits, our formalism leads to a generalization of the highly useful "softmax" scheme (Bridle, 1990; Rumelhart, 1989). This provides a deeper understanding of softmax and helps put our work in context.

The first factor in equation 5 is a perfectly well-defined function of $O$, but it could be impractical to evaluate it from its definition (summing over the calibration set) whenever it is needed. Therefore we sought a closed-form approximation for it. After making some ruthless approximations and carrying out the integration in equation 5, it reduces to

$$P(c|I) = \frac{\exp[T^\Delta(O_c - T^0)/\sigma_{cc}^2]}{\sum_{c'} \exp[T^\Delta(O_{c'} - T^0)/\sigma_{c'c'}^2]} \tag{6}$$

where $T^\Delta$ is the difference between the target values $(T^+ - T^-)$, $T^0$ is the average of the target values, and $\sigma_{cj}$ is the second moment of output unit $j$ for data in category $c$. This can be compared to the standard softmax expression

$$P(c|I) = \frac{\exp[\Gamma O_c]}{\sum_{c'} \exp[\Gamma O_{c'}]} \tag{7}$$

We see that our formula has three advantages: (1) it is clear how to handle the case where the targets are not symmetric about zero (non-vanishing $T^0$); (2) the "gain" of the exponentials depends on the category $c$; and (3) the gains can be calculated from measurable[1] properties of the data. Having the gain depend on the category makes a lot of sense; one can see in the figures that some categories

are more tightly clustered than others. One weakness that our equation 6 shares with softmax is the assumption that the output distribution of each output $j$ is circular (i.e. independent of $c$). This can be remedied by retracting some of the approximations leading to equation 6.

**Summary:** In a wide range of applications, it is extremely important to have good estimates of the probability of correct classification (as well as runner-up probabilities). We have shown how to create a network that computes the parameters of a probability distribution (or confidence interval) describing the set of outputs that are consistent with a given input and with the training data. The method has been described in terms of neural nets, but applies equally well to any parametric estimation technique that allows calculation of second derivatives. The analysis outlined here makes clear the assumptions inherent in previous schemes and offers a well-founded way of calculating the required probabilities.

## Footnotes

[1]Our formulas contain the overall confidence factor $\beta$, which is not as easily measurable as we would like.

# References

Becker, S. and Le Cun, Y. (1989). Improving the Convergence of Back-Propagation Learning with Second-Order Methods. In Touretzky, D., Hinton, G., and Sejnowski, T., editors, *Proc. of the 1988 Connectionist Models Summer School*, pages 29–37, San Mateo. Morgan Kaufman.

Bridle, J. S. (1990). Training Stochastic Model Recognition Algorithms as Networks can lead to Maximum Mutual Information Estimation of Parameters. In Touretzky, D., editor, *Advances in Neural Information Processing Systems*, volume 2, (Denver, 1989). Morgan Kaufman.

Denker, J. and leCun, Y. (1990). Transforming Neural-Net Output Levels to Probability Distributions. Technical Memorandum TM11359-901120-05, AT&T Bell Laboratories, Holmdel NJ 07733.

Denker, J., Schwartz, D., Wittner, B., Solla, S. A., Howard, R., Jackel, L., and Hopfield, J. (1987). Automatic Learning, Rule Extraction and Generalization. *Complex Systems*, 1:877–922.

Duda, R. and Hart, P. (1973). *Pattern Classification And Scene Analysis*. Wiley and Son.

Fogelman, F. (1990). personal communication.

Le Cun, Y., Boser, B., Denker, J. S., Henderson, D., Howard, R. E., Hubbard, W., and Jackel, L. D. (1990). Handwritten Digit Recognition with a Back-Propagation Network. In Touretzky, D., editor, *Advances in Neural Information Processing Systems*, volume 2, (Denver, 1989). Morgan Kaufman.

Le Cun, Y., Denker, J. S., and Solla, S. (1990). Optimal Brain Damage. In Touretzky, D., editor, *Advances in Neural Information Processing Systems*, volume 2, (Denver, 1989). Morgan Kaufman.

Rumelhart, D. E. (1989). personal communication.

Tishby, N., Levin, E., and Solla, S. A. (1989). Consistent Inference of Probabilities in Layered Networks: Predictions and Generalization. In *Proceedings of the International Joint Conference on Neural Networks*, Washington DC.

It is a pleasure to acknowledge useful conversations with John Bridle.